# Signal Detection in Noisy Weakly-Active Dendrites

**Amit Manwani and Christof Koch**
{quixote,koch}@klab.caltech.edu
Computation and Neural Systems Program
California Institute of Technology
Pasadena, CA 91125

## Abstract

Here we derive measures quantifying the information loss of a synaptic signal due to the presence of neuronal noise sources, as it electrotonically propagates along a weakly-active dendrite . We model the dendrite as an infinite linear cable, with noise sources distributed along its length. The noise sources we consider are thermal noise, channel noise arising from the stochastic nature of voltage-dependent ionic channels ($K^+$ and $Na^+$) and synaptic noise due to spontaneous background activity. We assess the efficacy of information transfer using a signal detection paradigm where the objective is to detect the presence/absence of a presynaptic spike from the post-synaptic membrane voltage. This allows us to analytically assess the role of each of these noise sources in information transfer. For our choice of parameters, we find that the synaptic noise is the dominant noise source which limits the maximum length over which information be reliably transmitted.

## 1 Introduction

This is a continuation of our efforts (Manwani and Koch, 1998) to understand the information capacity of a neuronal link (in terms of the specific nature of neural "hardware") by a systematic study of information processing at different biophysical stages in a model of a single neuron. Here we investigate how the presence of neuronal noise sources influences the information transmission capabilities of a simplified model of a weakly-active dendrite. The noise sources we include are, thermal noise, channel noise arising from the stochastic nature of voltage-dependent channels ($K^+$ and $Na^+$) and synaptic noise due to spontaneous background activity. We characterize the noise sources using analytical expressions of their current power spectral densities and compare their magnitudes for dendritic parameters reported in literature (Mainen and Sejnowski, 1998). To assess the role of these noise sources on dendritic integration, we consider a simplified scenario and model the dendrite as a lin-

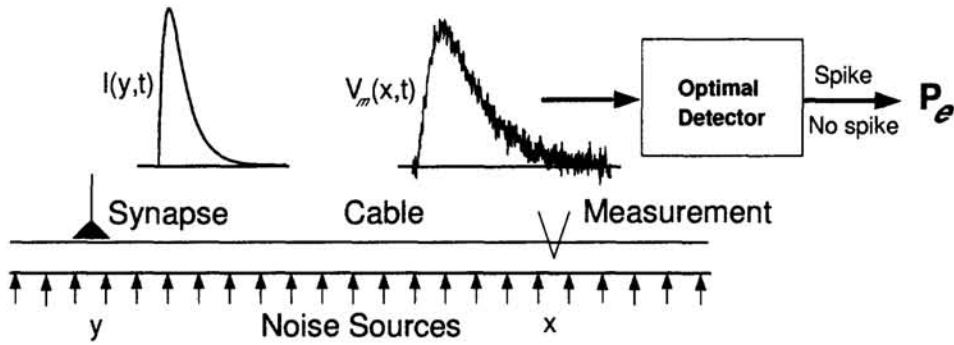

Figure 1: Schematic diagram of a simplified dendritic channel. The dendrite is modeled a weakly-active 1-D cable with noise sources distributed along its length. Loss of signal fidelity as it propagates from a synaptic location (input) $y$ to a measurement (output) location $x$ is studied using a signal detection task. The objective is to optimally detect the presence of the synaptic input $I(y, t)$ (in the form of a unitary synaptic event) on the basis of the noisy voltage waveform $V_m(x, t)$, filtered by the cable's Green's function and corrupted by the noise sources along the cable. The probability of error, $P_e$ is used to quantify task performance.

ear, infinite, one-dimensional cable with distributed current noises. When the noise sources are weak so that the corresponding voltage fluctuations are small, the membrane voltage satisfies a linear stochastic differential equation satisfied. Using linear cable theory, we express the power spectral density of the voltage noise in terms of the Green's function of an infinite cable and the current noise spectra. We use these results to quantify the efficacy of information transfer under a "signal detection" paradigm[1] where the objective is to detect the presence/absence of a presynaptic spike (in the form of an epsc) from the post-synaptic membrane voltage along the dendrite. The formalism used in this paper is summarized in **Figure 1**.

## 2 Neuronal Noise Sources

In this section we consider some current noise sources present in nerve membranes which distort a synaptic signal as it propagates along a dendrite. An excellent treatment of membrane noise is given in DeFelice (1981) and we refer the reader to it for details. For a linear one-dimensional cable, it is convenient to express quantities in specific length units. Thus, we express all conductances in units of S/$\mu$m and current power spectra in units of A$^2$/Hz $\mu$m.

### A. Thermal Noise
Thermal noise arises due to the random thermal agitation of electrical charges in a conductor and represents a fundamental lower limit of noise in a system. A conductor of resistance $R$ is equivalent to a noiseless resistor $R$ in series with a voltage noise source $V_{th}(t)$ of spectral density $S_{Vth}(f) = 2kTR$ (V$^2$/Hz), or a noiseless resistor $R$ in parallel with a current noise source, $I_{th}(t)$ of spectral density $S_{Ith}(f) = 2kT/R$ (A$^2$/ Hz), where $k$ is the Boltzmann constant and $T$ is the absolute temperature of the conductor[2]. The transverse resistance $r_m$ (units of $\Omega$ $\mu$m) of a nerve membrane is due to the combined resistance of the lipid bilayer and the resting conductances of various voltage-gated, ligand-gated and leak channels embedded in the lipid matrix. Thus, the current noise due to $r_m$, has power

spectral density,

$$S_{Ith}(f) = \frac{2kT}{r_m} \tag{1}$$

## B. Channel Noise

Neuronal membranes contain microscopic voltage-gated and ligand-gated channels which open and close randomly. These random fluctuations in the number of channels is another source of membrane noise. We restrict ourselves to voltage-gated $K^+$ and $Na^+$ channels, although the following can be used to characterize noise due to other types of ionic channels as well. In the classical Hodgkin-Huxley formalism (Koch, 1998), a $K^+$ channel consists of four identical two-state sub-units (denoted by $n$) which can either be open or closed. The $K^+$ channel conducts only when all the sub-units are in their open states. Since the sub-units are identical, the channel can be in one of five states; from the state in which all the sub-units are closed to the open state in which all sub-units are open. Fluctuations in the number of open channels cause a random $K^+$ current $I_K$ of power spectral density (DeFelice, 1981)

$$S_{IK}(f) = \eta_K \gamma_K^2 (V_m - E_K)^2 n_\infty^4 \sum_{i=1}^{4} \binom{4}{i} (1 - n_\infty)^i n_\infty^{4-i} \frac{2\,\theta_n/i}{1 + 4\pi^2 f^2 (\theta_n/i)^2}. \tag{2}$$

where $\eta_K$, $\gamma_K$ and $E_K$ denote the $K^+$ channel density (per unit length), the $K^+$ single channel conductance and the $K^+$ reversal potential respectively. Here we assume that the membrane voltage has been clamped to a value $V_m$. $n_\infty$ and $\theta_n$ are the steady-state open probability and relaxation time constant of a single $K^+$ sub-unit respectively and are in general non-linear functions of $V_m$ (Koch, 1998). When $V_m$ is close to the resting potential $V_{rest}$ (usually between -70 to -65 mV), $n_\infty \ll 1$ and one can simplify $S_{IK}(f)$ as

$$S_{IK}(f) \approx \eta_K \gamma_K^2 (V_{rest} - E_K)^2 n_\infty^4 (1 - n_\infty)^4 \frac{2\,\theta_n/4}{1 + 4\pi^2 f^2 (\theta_n/4)^2} \tag{3}$$

Similarly, the Hodgkin-Huxley $Na^+$ channel is characterized by three identical activation sub-units (denoted by $m$) and an inactivation sub-unit (denoted by $h$). The $Na^+$ channel conducts only when all the $m$ sub-units are open and the $h$ sub-unit is not inactivated. Thus, the $Na^+$ channel can be in one of eight states from the state corresponding to all $m$ sub-units closed and the $h$ sub-unit inactivated to the open state with all $m$ sub-units open and the $h$ sub-unit not inactivated. $m_\infty$ (resp. $h_\infty$) and $\theta_m$ (resp. $\theta_h$) are the corresponding steady-state open probability and relaxation time constant of a single $Na^+$ $m$ (resp. $h$) sub-unit respectively. For $V_m \approx V_{rest}$, $m_\infty \ll 1$, $h_\infty \approx 1$ and

$$S_{INa}(f) \approx \eta_{Na} \gamma_{Na}^2 (V_{rest} - E_{Na})^2 m_\infty^3 (1 - m_\infty)^3 h_\infty^2 \frac{2\,\theta_m/3}{1 + 4\pi^2 f^2 (\theta_m/3)^2} \tag{4}$$

where $\eta_{Na}$, $\gamma_{Na}$ and $E_{Na}$ denote the $Na^+$ channel density, the $Na^+$ single channel conductance and the sodium reversal potential respectively.

## C. Synaptic Noise

In addition to voltage-gated ionic channels, dendrites are also awash in ligand-gated synaptic receptors. We restrict our attention to fast voltage-independent (AMPA-like) synapses. A commonly used function to represent the postsynaptic conductance change in response to a presynaptic spike is the *alpha* function (Koch, 1998)

$$g_\alpha(t) = \frac{g_{peak}\,e}{t_{peak}}\,t\,e^{-t/t_{peak}},\ 0 \le t < \infty \tag{5}$$

where $g_{peak}$ denotes the peak conductance change and $t_{peak}$ the time-to-peak of the conductance change. We shall assume that for a spike train $s(t) = \sum_j \delta(t - t_j)$, the postsynaptic conductance is given $g_{Syn}(t) = \sum_j g_\alpha(t - t_j)$. This ignores inter-spike interaction

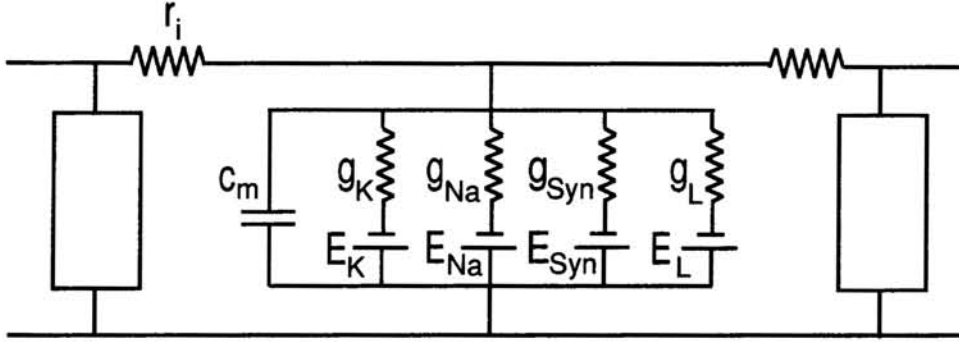

Figure 2: Schematic diagram of the equivalent electrical circuit of a linear dendritic cable. The dendrite is modeled as an infinite ladder network. $r_i$ (units of $\Omega/\mu$ m) denotes the longitudinal cytoplasmic resistance; $c_m$ (units of F/$\mu$ m) and $g_L$ (units of S/$\mu$ m) denote the transverse membrane capacitance and conductance (due to leak channels with reversal potential $E_L$) respectively. The membrane also contains active channels ($K^+$, $Na^+$) with conductances and reversal potentials denoted by ($g_K$, $g_{Na}$) and ($E_K$, $E_{Na}$) respectively, and fast voltage-independent (AMPA-like) synapses with conductance $g_{Syn}$ and reversal potential $E_{Syn}$.

and synaptic saturation. The synaptic current is given by $i_{Syn}(t) = g_{Syn}(t)(V_m - E_{Syn})$ where $E_{Syn}$ is the synaptic reversal potential. If the spike train can be modeled as a homogeneous Poisson process with mean firing rate $\lambda_n$, the power spectrum of $i_{Syn}(t)$ can be computed using Campbell's theorem (Papoulis, 1991)

$$S_{ISyn}(f) = \eta_{Syn}\lambda_n(V_m - E_{Syn})^2 \mid G_\alpha(f) \mid^2 , \qquad (6)$$

where $\eta_{Syn}$ denotes the synaptic density and $G_\alpha(f) = \int_0^\infty g_\alpha(t) \, exp(-j2\pi ft) \, dt$ is the Fourier transform of $g_\alpha(t)$. Substituting for $g_\alpha(t)$ gives

$$S_{ISyn}(f) = \eta_{Syn} \, \lambda_n \, \frac{(e \, g_{peak}t_{peak}(V_m - E_{Syn}))^2}{(1 + 4\pi^2 f^2 t_s^2)^2} \qquad (7)$$

## 3   Noise in Linear Cables

The linear infinite cable corresponding to a dendrite is modeled by the ladder network shown in **Figure 2**. The membrane voltage $V_m(x,t)$ satisfies the differential equation (Tuckwell, 1988),

$$\frac{\partial^2 V_m}{\partial^2 x} = r_i \left[ c_m \frac{\partial V_m}{\partial t} + g_K(V_m - E_K) + g_{Na}(V_m - E_{Na}) \right.$$
$$\left. + g_{Syn}(V_m - E_{Syn}) + g_L(V_m - E_L) \right] \qquad (8)$$

Since the ionic conductances are random and nonlinearly related to $V_m$, eq. 8 is a nonlinear stochastic differential equation. If the voltage fluctuations (denoted by $V$) around the resting potential $V_{rest}$ are small, one can express the conductances as small deviations (denoted by $\tilde{g}$) from their corresponding resting values and transform eq. 8 to

$$-\lambda^2 \frac{\partial^2 V(x,t)}{\partial x^2} + \tau \frac{\partial V(x,t)}{\partial t} + (1 + \delta)V(x,t) = \frac{I_n}{G} \qquad (9)$$

where $\lambda^2 = 1/(r_i G)$ and $\tau = c_m/G$ denote the length and time constant of the membrane respectively. $G$ is the passive membrane conductance and is given by the sum of the resting values of all the conductances. $\delta = \tilde{g}_K + \tilde{g}_{Na} + \tilde{g}_{Syn}/G$ represents the random changes in the membrane conductance due to synaptic and channel stochasticity; $\delta$

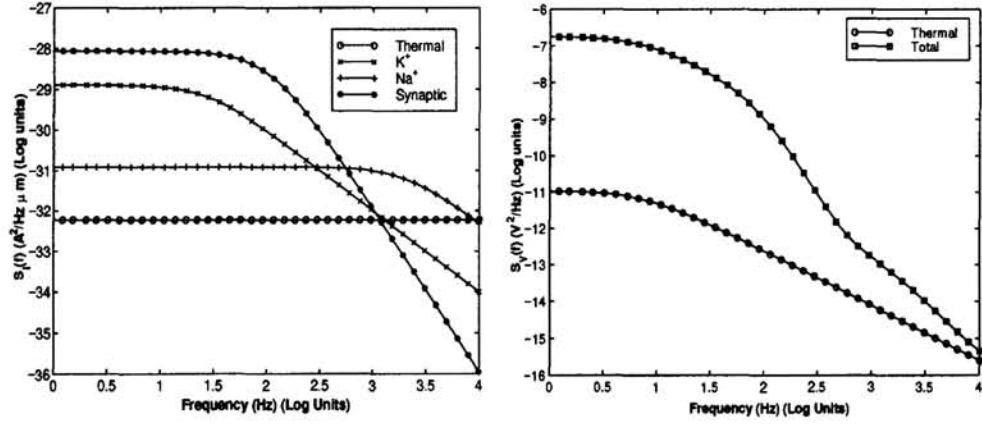

Figure 3: **(a)** Comparison of current spectra $S_I(f)$ of the four noise sources we consider. Synaptic noise is the most dominant source source of noise and thermal noise, the smallest. **(b)** Voltage noise spectrum of a 1-D infinite cable due to the current noise sources. $S_{Vth}(f)$ is also shown for comparison. Summary of the parameters used (adopted from Mainen and Sejnowski, 1998) : $R_m = 40$ k$\Omega$cm$^2$, $C_m = 0.75$ $\mu$F/cm$^2$, $r_i = 200$ $\Omega$cm, d (dend. dia.) $= 0.75$ $\mu$m, $\eta_K = 2.3$ $\mu$m$^{-1}$, $\eta_{Na} = 3$ $\mu$m$^{-1}$, $\eta_{Syn} = 0.1$ $\mu$m$^{-1}$, $E_K = -95$ mV, $E_{Na} = 50$ mV, $E_{Syn} = 0$ mV, $E_L = V_{rest} = 70$ mV, $\gamma_K = \gamma_{Na} = 20$ pS.

$V_{rest}) + I_{th}$ denotes the total effective current noise due to the different noise sources. In order to derive analytical closed-form solutions to eq. 9, we further assume that $\delta << 1$[3], which reduces it to the familiar one-dimensional cable equation with noisy current input (Tuckwell, 1988). For resting initial conditions (no charge stored on the membrane at $t = 0$), $V$ is linearly related to $I_n$ and can be obtained by convolving $I_n$ with the Green's function $g(x, y, t)$ of the cable for the appropriate boundary conditions. It has been shown that $V(x, t)$ is an asymptotically wide-sense stationary process (Tuckwell and Walsh, 1991) and its power spectrum $S_V(x, f)$ can be expressed in terms of the power spectrum of $I_n$, $S_n(f)$ as

$$S_V(x, f) = \frac{S_n(f)}{G^2} \int_{-\infty}^{\infty} |\mathcal{G}(x, x', f)|^2 \, dx' \qquad (10)$$

where $\mathcal{G}(x, x', f)$ is the Fourier transform of $g(x, x', t)$. For an infinite cable

$$g(X, X', T) = \frac{e^{-T}}{\sqrt{4\pi T}} e^{\frac{-(X-X')^2}{4T}}, \quad -\infty < X, X' < \infty, \; 0 \leq T < \infty \qquad (11)$$

where $X = x/\lambda$, $X' = x'/\lambda$ and $T = t/\tau$ are the corresponding dimensionless variables. Substituting for $g(x, x', t)$ we obtain

$$S_V(f) = \frac{S_n(f)}{2\lambda G^2} \frac{\sin\left(\frac{\tan^{-1}(2\pi f\tau)}{2}\right)}{2\pi f\tau \, (1 + (2\pi f\tau)^2)^{1/4}} \qquad (12)$$

Since the noise sources are independent, $S_n(f) = S_{Ith}(f) + S_{IK}(f) + S_{INa}(f) + S_{ISyn}(f)$. Thus, eq. 12 allows us to compute the relative contribution of each of the noise sources to the voltage noise. The current and voltage noise spectra for biophysically relevant parameter values (Mainen and Sejnowski, 1998) are shown in **Figure 3**.

## 4   Signal Detection

The framework and notation used here are identical to that in Manwani and Koch (1998) and so we refer the reader to it for details. The goal in the signal detection task is to optimally decide between the two hypotheses

$$
\begin{aligned}
H_0 &: \quad y(t) = n(t), & 0 \le t \le T & \qquad \underline{\text{Noise}} \\
H_1 &: \quad y(t) = g(t) * s(t) + n(t), & 0 \le t \le T & \qquad \underline{\text{Signal} + \text{Noise}} \quad (13)
\end{aligned}
$$

where $n(t)$, $g(t)$ and $s(t)$ denote the dendritic voltage noise, the Green's function of the cable (function of the distance between the input and measurement locations) and the epsc waveform (due to a presynaptic spike) respectively. The decision strategy which minimizes the probability of error $P_e = p_0 P_f + p_1 P_m$, where $p_0$ and $p_1 = (1 - p_0)$ are the prior probabilities of $H_0$ and $H_1$ respectively, is

$$
\Lambda(y) \underset{H_0}{\overset{H_1}{\gtrless}} \mathcal{L}_0 \qquad (14)
$$

where $\Lambda(y) = P[y|H_1]/P[y|H_0]$ and $\mathcal{L}_0 = p_0/(1 - p_0)$. $P_f$ and $P_m$ denote the false alarm and miss probability respectively. Since $n(t)$ arises due to the effect of several independent noise sources, by invoking the Central Limit theorem, we can assume that $n(t)$ is Gaussian, for which eq. 14 reduces to $r \underset{H_0}{\overset{H_1}{\gtrless}} \eta$. $r = \int_0^\infty y(t)\, h_d(-t)\, dt$ is a correlation between $y(t)$ and the *matched filter* $h_d(t)$, given in the Fourier domain as $H_d(f) = e^{-j2\pi f T} \mathcal{G}^*(f) S^*(f)/S_n(f)$. $\mathcal{G}(f)$ and $S(f)$ are Fourier transforms of $g(t)$ and $s(t)$ respectively and $S_n(f)$ is the noise power spectrum. The conditional means and variances of the Gaussian variable $r$ under $H_0$ and $H_1$ are $\mu_0 = 0, \mu_1 = \int_{-\infty}^\infty |G(f)S(f)|^2/S_n(f)\, df$ and $\sigma_0^2 = \sigma_1^2 = \sigma^2 = \mu_1$ respectively. The error probabilities are given by $P_f = \int_\eta^\infty P[r|H_0]\, dr$ and $P_m = \int_{-\infty}^\eta P[r|H_1]\, dr$. The optimal value of the threshold $\eta$ depends on $\sigma$ and the prior probability $p_0$. For equiprobable hypotheses ($p_0 = 1 - p_0 = 0.5$), the optimal $\eta = (\mu_0 + \mu_1)/2 = \sigma^2/2$ and $P_e = 0.5\, \mathrm{Erfc}[\sigma/2\sqrt{2}]$. One can also regard the overall decision system as an effective binary channel. Let $M$ and $D$ be binary variables which take values in the set $\{H_0, H_1\}$ and denote the input and output of the dendritic channel respectively. Thus, the system performance can equivalently be assessed by computing the mutual information between $M$ and $D$, $I(M;D) = \mathcal{H}(p_o (1 - P_m) + (1 - p_o) P_f) - p_o \mathcal{H}(P_m) - (1 - p_o)\mathcal{H}(P_f$ (Cover and Thomas, 1991) where $\mathcal{H}(x)$ is the binary entropy function. For equi-probable hypotheses, $I(M;D) = 1 - \mathcal{H}(P_e)$ bits. It is clear from the plots for $P_e$ and $I(M;D)$ (**Figure 4**) as a function of the distance between the synaptic (input) and the measurement (output) location that an epsc. can be detected with almost certainty at short distances, after which, there is a rapid decrease in detectability with distance. Thus, we find that membrane noise may limit the maximum length of a dendrite over which information can be transmitted reliably.

## 5   Conclusions

In this study we have investigated how neuronal noise sources might influence and limit the ability of one-dimensional cable structures to propagate information. When extended to realistic dendritic geometries, this approach can help address questions as, is the length of the apical dendrite in a neocortical pyramidal cell limited by considerations of signal-to-noise, which synaptic locations on a dendritic tree (if any) are better at transmitting information, what is the functional significance of active dendrites (Yuste and Tank, 1996) and so on. Given the recent interest in dendritic properties, it seems timely to apply an information-theoretic approach to study dendritic integration. In an attempt to experimentally verify

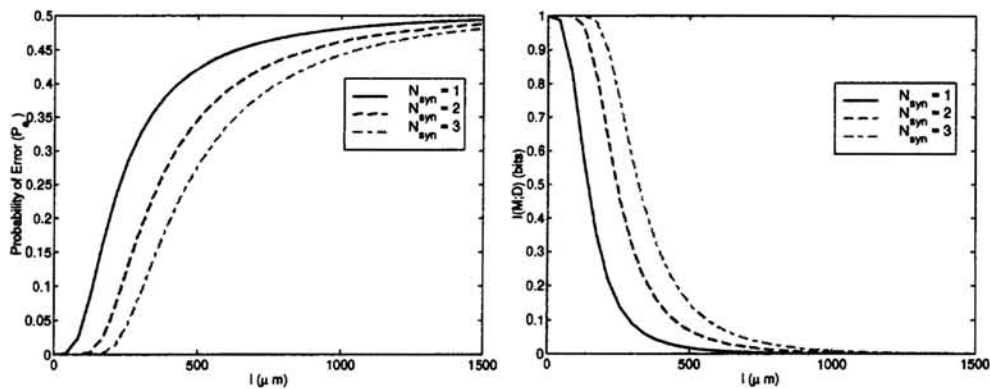

Figure 4: Information loss in signal detection. (**a**) Probability of Error ($P_e$) and (**b**) Mutual information ($I(M;D)$) for an infinite cable as a function of distance from the synaptic input location. Almost perfect detection occurs for small distances but performance degrades steeply over larger distances as the signal-to-noise ratio drops below some threshold. This suggests that dendritic lengths may be ultimately limited by signal-to-noise considerations. Epsc. parameters: $g_{peak}$ = 0.1 nS, $t_{peak}$ = 1.5 msec and $E_{Syn}$ = 0 mV. $N_{syn}$ is the number of synchronous synapses which activate in response to a pre-synaptic action potential.

the validity of our results, we are currently engaged in a quantitative comparison using neocortical pyramidal cells (Manwani *et al*, 1998).

## Acknowledgements

This research was supported by NSF, NIMH and the Sloan Center for Theoretical Neuroscience. We thank Idan Segev, Elad Schneidman, Miki London, Yosef Yarom and Fabrizio Gabbiani for illuminating discussions.

## Footnotes

[1]For sake of brevity, we do not discuss the corresponding signal estimation paradigm as in Manwani and Koch (1998).

[2]Since the power spectra of real signals are even functions of frequency, we choose the double-sided convention for all power spectral densities.

[3]Using self-consistency, we find the assumption to be satisfied in our case. In general, it needs verified on a case-by-case basis.

## References

DeFelice, L.J. (1981) *Membrane Noise*. New York: Plenum Press.

Cover, T.M., and Thomas, J.A. (1991) *Elements of Information Theory*. New York: Wiley.

Koch, C. (1998) *Biophysics of Computation: Information Processing in Single Neurons*. Oxford University Press.

Mainen, Z.F. and Sejnowski, T.J. (1998) "Modeling active dendritic processes in pyramidal neurons," In: *Methods in Neuronal Modeling: From Ions to Networks*, Koch, C. and Segev, I., eds., Cambridge: MIT Press.

Manwani, A. and Koch, C. (1998) "Synaptic transmission: An information-theoretic perspective," In: Kearns, M., Jordan, M. and Solla, S., eds., *Advances in Neural Information Processing Systems,"* Cambridge: MIT Press.

Manwani, A., Segev, I., Yarom, Y and Koch, C. (1998) "Neuronal noise sources in membrane patches and linear cables," In: *Soc. Neurosci. Abstr.*

Papoulis, A. (1991) *Probability, Random Variables and Stochastic Processes*. New York: McGraw-Hill.

Tuckwell, H.C. (1988) *Introduction to Theoretical Neurobiology: I*. New York: Cambridge University Press.

Tuckwell, H.C. and Walsh, J.B. (1983) "Random currents through nerve membranes I. Uniform poisson or white noise current in one-dimensional cables," *Biol. Cybern.* **49**:99-110.

Yuste, R. and Tank, D.W. (1996) "Dendritic integration in mammalian neurons, a century after Cajal,"